# FEEDBACK SYNAPSE TO CONE AND LIGHT ADAPTATION

Josef Skrzypek
Machine Perception Laboratory
UCLA - Los Angeles, California 90024
INTERNET: SKRZYPEK@CS.UCLA.EDU

## Abstract

Light adaptation (LA) allows cone vision to remain functional between twilight and the brightest time of day even though, at any one time, their intensity-response (I-R) characteristic is limited to 3 log units of the stimulating light. One mechanism underlying LA, was localized in the outer segment of an isolated cone (1,2). We found that by adding annular illumination, an I-R characteristic of a cone can be shifted along the intensity domain. Neural network involving feedback synapse from horizontal cells to cones is involved to be in register with ambient light level of the periphery. An equivalent electrical circuit with three different transmembrane channels leakage, photocurrent and feedback was used to model static behavior of a cone. SPICE simulation showed that interactions between feedback synapse and the light sensitive conductance in the outer segment can shift the I-R curves along the intensity domain, provided that phototransduction mechanism is not saturated during maximally hyperpolarized light response.

## 1 INTRODUCTION

### 1.1 Light response in cones

In the vertebrate retina, cones respond to a small spot of light with sustained hyperpolarization which is graded with the stimulus over three log units of intensity [5]. Mechanisms underlying this I-R relation was suggested to result from statistical superposition of invariant single-photon, hyperpolarizing responses involvnig sodium conductance changes that are gated by cyclic nucleotides (see 6). The shape of the response measured in cones depends on the size of the stimulating spot of light, presumably because of peripheral signals mediated by a negative feedback synapse from horizontal cells [7,8]; the hyperpolarizing response to the spot illumination in the central portion of the cone receptive field is antagonized by light in the surrounding periphery [11,12,13]. Thus the cone

membrane is influenced by two antagonistic effects; 1) feedback, driven by peripheral illumination and 2) the light sensitive conductance, in the cone outer segment. Although it has been shown that key aspects of adaptation can be observed in isolated cones [1,2,3], the effects of peripheral illumination on adaptation as related to feedback input from horizontal cells have not been examined. It was reported that under appropriate stimulus conditions the resting membrane potential for a cone can be reached at two drastically different intensities for a spot/annulus combinations [8,14].

We present here experimental data and modeling results which suggests that results of feedback from horizontal cells to cones resemble the effect of the neural component of light adaptation in cones. Specifically, peripheral signals mediated via feedback synapse reset the cone sensitivity by instantaneously shifting the I-R curves to a new intensity domain. The full range of light response potentials is preserved without noticeable compression.

## 2  RESULTS

### 2.1  Identification of cones

Preparation and the general experimental procedure as well as criteria for identification of cones has been detailed in [15,8]. Several criteria were used to distinguish cones from other cells in the OPL such as: 1) the depth of recording in the retina [11, 13], 2) the sequence of penetrations concomitant with characteristic light responses, 3) spectral response curves [18], 4) receptive field diameter [8], 5) the fastest time from dark potential to the peak of the light response [8, 15], 6)domain of I-R curves and 7) staining with Lucipher Yellow [8, 11, 13]. These values represent averages derived from all intracellular recordings in 37 cones, 84 bipolar cells, more than 1000 horizontal cells, and more than 100 rods.

### 2.2  Experimental procedure

After identifying a cone, its I-R curve was recorded. Then, in a presence of center illumination (diameter = 100 um) which elicited maximal hyperpolarization from a cone, the periphery of the receptive field was stimulated with an annulus of inner diameter (ID) = 750 um and the outer diameter (OD) = 1500 um. The annular intensity was adjusted to elicit depolarization of the membrane back to the dark potential level. Finally, the center intensity was increased again in a stepwise manner to antagonize the effect of peripheral illumination, and this new I-R curve was recorded.

### 2.3  Peripheral illumination shifts the I-R curve in cones

Sustained illumination of a cone with a small spot of light, evokes a hyperpolarizing response, which after transient peak gradually repolarizes to some steady level (Fig. 1a). When the periphery of the retina is illuminated with a ring of light in the presence of center spot, the antagonistic component of response can be recorded in a form of sustained depolarization. It has been argued previously that in the tiger salamander cones, this type of response in cones is mediated via synaptic input from horizontal cells. [11, 12].

The significance of this result is that the resting membrane potential for this cone can be reached at two drastically different intensities for a spot/annulus combinations; The action of an annular illumination is a fast depolarization of the membrane; the whole process is completed in a fraction of a second unlike the previous reports where the course of light-adaptation lasted for seconds or even minutes.

Response due to spot of light measured at the peak of hyperpolarization, increased in magnitude with increasing intensity over three log units (fig. 1.a). The same data is plotted as open circles in fig. 1.b. Initially, annulus presented during the central illumination did not produce a noticeable response. Its amplitude reached maximum when the center spot intensity was increased to 3 log units. Further increase of center intensity resulted in disappearance of the annulus- elicited depolarization. Feedback action is graded with annular intensity and it depends on the balance between amount of light falling on the center and the surround of the cone receptive field. The change in cone's membrane potential, due to combined effects of central and annular illumination is plotted as filled circles in fig. 1b. This new intensity-response curve is shifted along the intensity axis by approximately two log units. Both I-R curves span approximately three log units of intensity. The I-R curve due to combined center and surround illumination can be described by the function $V/Vm = I/(I+k)$ [16] where Vm is a peak hyperpolarization and k is a constant intensity generating half-maximal response. This relationship $[x/(x+k)]$ was suggested to be an indication of the light adaptation [2]. The I-R curve plotted using peak response values (open circles), fits a continuous line drawn according to equation $(1-exp(-kx))$. This has been argued previously to indicate absence of light adaptation [2,1]. There is little if any compression or change in gain after the shift of the cone operating point to some new domain of intensity. The results suggest that peripheral illumination can shift the center-spot elicited I-R curve of the cone thus resetting the response-generating mechanism in cones.

## 2.4    Simulation of a cone model

The results presented in the previous sections imply that maximal hyperpolarization for the cone membrane is not limited by the saturation in the phototransduction process alone. It seems reasonable to assume that such a limit may be in part determined by the batteries of involved ions. Furthermore, it appears that shifting I-R curves along the intensity domain is not dependent solely on the light adaptation mechanism localized to the outer segment of a cone. To test these propositions we developed a simplified compartmental model of a cone (Fig.2.) and we exercised it using SPICE (Vladimirescu et al., 1981).

All interactions can be modeled using Kirchoff's current law; membrane current is $c_{m(dV/dt)}+I_{ionic}$. The leakage current is $I_{leak} = G_{leak}(V_m-E_{leak})$, light sensitive current is $I_{light} = G_{light}*(V_m-E_{light})$ and the feedback current is $I_{fb} = G_{fb}*(V_m-E_{fb})$. The left branch represents ohmic leakage channels ($G_{leak}$) which are associated with a constant battery $E_{leak}$ ( -70 mV). The middle branch represents the light sensitive conductance ($G_{light}$) in series with +1 mV ionic battery ($E_{light}$) [18]. Light adaptation effects could be incorporated here by making Glight time varying and dependent on internal concentration of Calcium ions. In our preliminary studies we were only interested in examining whether the shift of I-R is possible and if it would explain the disappearance of depolarizing FB response with hyperpolarization by the center light. This can be done with passive measurements of membrane potential amplitude. The right-most branch represents ionic channels that are controlled by the feedback synapse. With, $E_{fb}$ = -65 mV [11] $G_{fb}$ is a time and voltage independent feedback conductance.

The input resistance of an isolated cone is taken to be near 500 Mohm (270 Mohm Attwell, et al., 82). Assuming specific membrane resistance of 5000 Ohm*cm*cm and that a cone is 40 microns long and has a 8 micron diameter at the base we get the leakage conductance $G_{leak}$ = 1/(1Gohm). In our studies we assume $G_{leak}$ to be linear altghouth there is evidence that cone membrane rectifies (Skrzypek, 79). The $G_{light}$ and $G_{fb}$ are assumed to be equal and add up to 1/(1Gohm). The Glight varies with light intensity in proportion of two to three log units of intensity for a tenfold change in conductance. This relation was derived empirically, by comparing intensity response data obtained from a cone $\{V_m=f(LogI)\}$ to $\{V_m=f(LogG_{light})\}$ generated by the model. The changes in Gfb have not been calibrated to changes in light intensity of the annulus. However, we assume that Gfb can not undergo variation larger that $G_{light}$.

Figure 3 shows the membrane potential changes generated by the model plotted as a function of $R_{light}$, at different settings of the "feedback" resistance $R_{fb}$. With increasing $R_{fb}$, there is a parallel shift along the abscissa without any changes in the shape of the curve. Increase in $R_{light}$ corresponds to increase in light intensity and the increasing magnitude of the light response from 0mV ($E_{light}$) all the way down to -65 mV ($E_{fb}$). The increase in $R_{fb}$ is associated with increasing intensity of the annular illumination, which causes additional hyperpolarization of the horizontal cell and consequently a decrease in "feedback" transmitter released from HC to cones. Since we assume the $E_{fb}$ =−65mV, a more negative level than the normal resting membrane potential, a decrease in $G_{fb}$ would cause a depolarizing response in the cone. This can be observed here as a shift of the curve along the abscissa. In our model, a hundred fold change in feedback resistance from 0.01Gohm to 1Gohm, resulted in shift of the "response-intensity" curve by approximately two log units along the abscissa. The relationship between changes in $R_{fb}$ and the shift of the "response-intensity" curve is nonlinear and additional increases in Rfb from 1Gohm to 100Gohm results in decreasing shifts.

Membrane current undergoes similar parallel shift with changes in feedback conductance. However, the photocurrent ($I_{light}$) and the feedback current ($I_{fb}$), show only saturation with increasing $G_{light}$ (not shown). The limits of either $I_{light}$ or $I_{fb}$ currents are defined by the batteries of the model. Since these currents are associated with batteries of opposite polarities, the difference between them at various settings of the feedback conductance $G_{fb}$ determines the amount of shift for $I_{leak}$ along the abscissa. The compression in shift of "response intensity" curves at smaller values of $G_{fb}$ results from smaller and smaller current flowing through the feedback branch of the circuit. Consequently, a smaller $G_{fb}$ changes are required to get response in the dark than in the light.

The shifting of the "response-intensity" curves generated by our model is not due to light adaptation as described by [1,2] although it is possible that feedback effects could be involved in modulating light-sensitive channels. Our model suggests that in order to generate additional light response after the membrane of a cone was fully hyperpolarized by light, it is insufficient to have a feedback effect alone that would depolarize the cone membrane. Light sensitive channels that were not previously closed [18] must also be available.

# 3  DISCUSSION

The results presented here suggest that synaptic feedback from horizontal cells to cones could contribute to the process of light adaptation at the photoreceptor level. A complete explanation of the underlying mechanism requires further studies but the results seem to suggest that depolarization of the cone membrane by a peripheral illumination, resets the response-generating process in the cone. This result can be explained withing the framework of the current hypothesis of the light adaptation, recently summarized by [6].

It is conceivable that feedback transmitter released from horizontal cells in the dark, opens channels to ions with reversal potential near -65 mV [11]. Hence, hyperpolarizing cone membrane by increasing center spot intensity would reduce the depolarizing feedback response as cone nears the battery of involved ions. Additional increase in annular illumination, further reduces the feedback transmitter and the associated feedback conductance thus pushing cone's membrane potential away from the "feedback" battery. Eventually, at some values of the center intensity, cone membrane is so close to -65 mV that no change in feedback conductance can produce a depolarizing response.

## ACKNOWLEDGEMENTS

Special gratitude to Prof. Werblin for providing a superb research environment and generous support during early part of this project. We acknowledge partial support by NSF grant ECS-8307553, ARCO-UCLA Grant #1, UCLA-SEASNET Grant KF-21, MICRO-Hughes grant #541122-57442, ONR grant #N00014-86-K-0395, ARO grant DAAL03-88- K-0052

## REFERENCES

1. Nakatani, K., & Yau, K.W. (1988). Calcium and light adaptation in retinal rods and cones. Nature. 334, 69-71.

2. Matthews, H.R., Murphy, R.L.W., Fain, G.L., & Lamb, T.D. (1988). Photoreceptor light adaptation is mediated by cytoplasmic calcium concentration. Nature, 334, 67-69.

3. Normann, R.A. & Werblin, F.S. (1974). Control of retinal sensitivity. I. Light and dark-adaptation of vertebrate rods and cones. J. Physiol. 63, 37-61.

4. Werblin, F.S. & Dowling, J.E. (1969). Organization of the retina of the mudpuppy, Necturus maculosus. II. Intracellular recording. J. Neurophysiol. 32, (1969),315-338.

5. Pugh, E.N. & Altman, J. Role for calcium in adaptation. Nature 334, (1988), 16-17.

6. O'Bryan P.M., Properties of the dpolarizing synaptic potential evoked by peripheral illumination in cones of the turtle retina. J.Physiol. Lond. 253, (1973), 207-223.

7. Skrzypek J., Ph.D. Thesis, University of California at Berkeley, (1979).

8. Skrzypek, J. & Werblin, F.S.,(1983). Lateral interactions in absence of feedback to cones. J. Neurophysiol. 49, (1983), 1007-1016.

9. Skrzypek, J. & Werblin, F.S., All horizontal cells have center-surround antagonistic receptive fields. ARVO Abstr., (1978).

10. Lasansky, A. Synaptic action mediating cone responses to annular illumination in the retina of the larval tiger salamander. J. Physiol. Lond. 310, (1981), 206-214.

11. Skrzypek J., Electrical coupling between horizontal vell bodies in the tiger salamander retina. Vision Res. 24, (1984), 701-711.

12. Naka, K.I. & Rushton, W.A.H. (1967). The generation and spread of S-potentials in fish (Cyprinidae) J. Physiol., 192, (1967), 437-461.

13. Attwell, D., Werblin, F.S. & Wilson, M. (1982a). The properties of single cones isolated from the tiger salamander retina. J. Physiol. 328, 259-283.

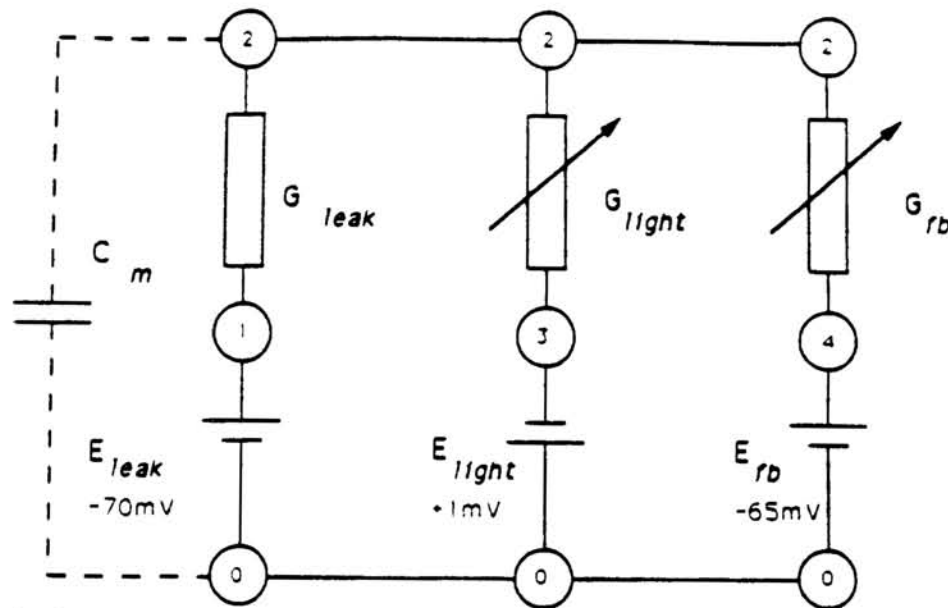

Fig. 2 Equivalent circuit model of a cone based on three different transmembrane chan­nels. The ohmic leakage channel consists of a constant conductance $G_{leak}$ in series with constant battery $E_{leak}$. Light sensitive channels are represented in the middle branch by $G_{light}$. Battery $E_{light}$, represents the reversal potential for light response at approximately OmV. Feedback synapse is shown in the right-most branch as a series combination of $G_{fb}$ and the battery $E_{fb} = -65mV$, representing reversal potential for annulus elicited, depolarizing response measured in a cone.

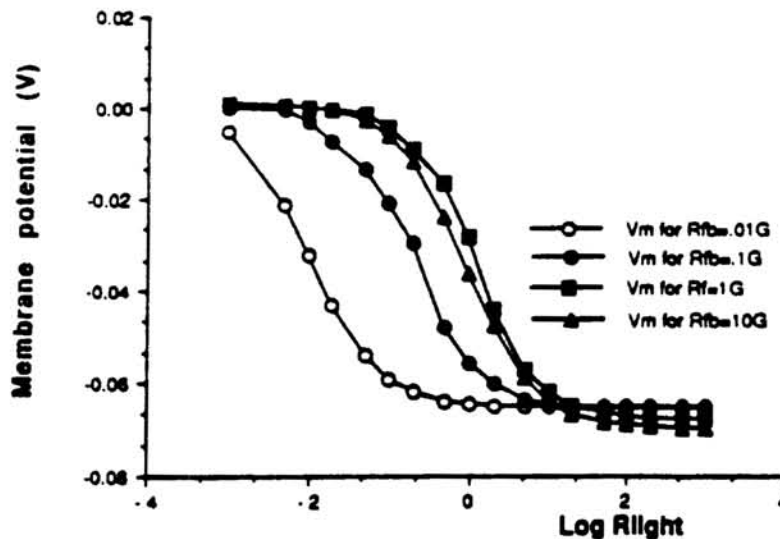

Fig. 3 Plot of the membrane potential versus the logarithm of light-sensitive resistance. The data was synthesized with the cone model simulated by SPICE. Both current and voltage curves can be fitted by x/(x+k) relation (not shown) at all different settings of Gfb (Rfb) indicated in the legend. The shift of the curves, measured at 1/2 maximal value (k=x) spans about two log units. With increasing settings of Rfb (10 Gohms), curves be­gin to cross (Vm at -65mV) signifying decreasing contribution of "feedback" synapse.

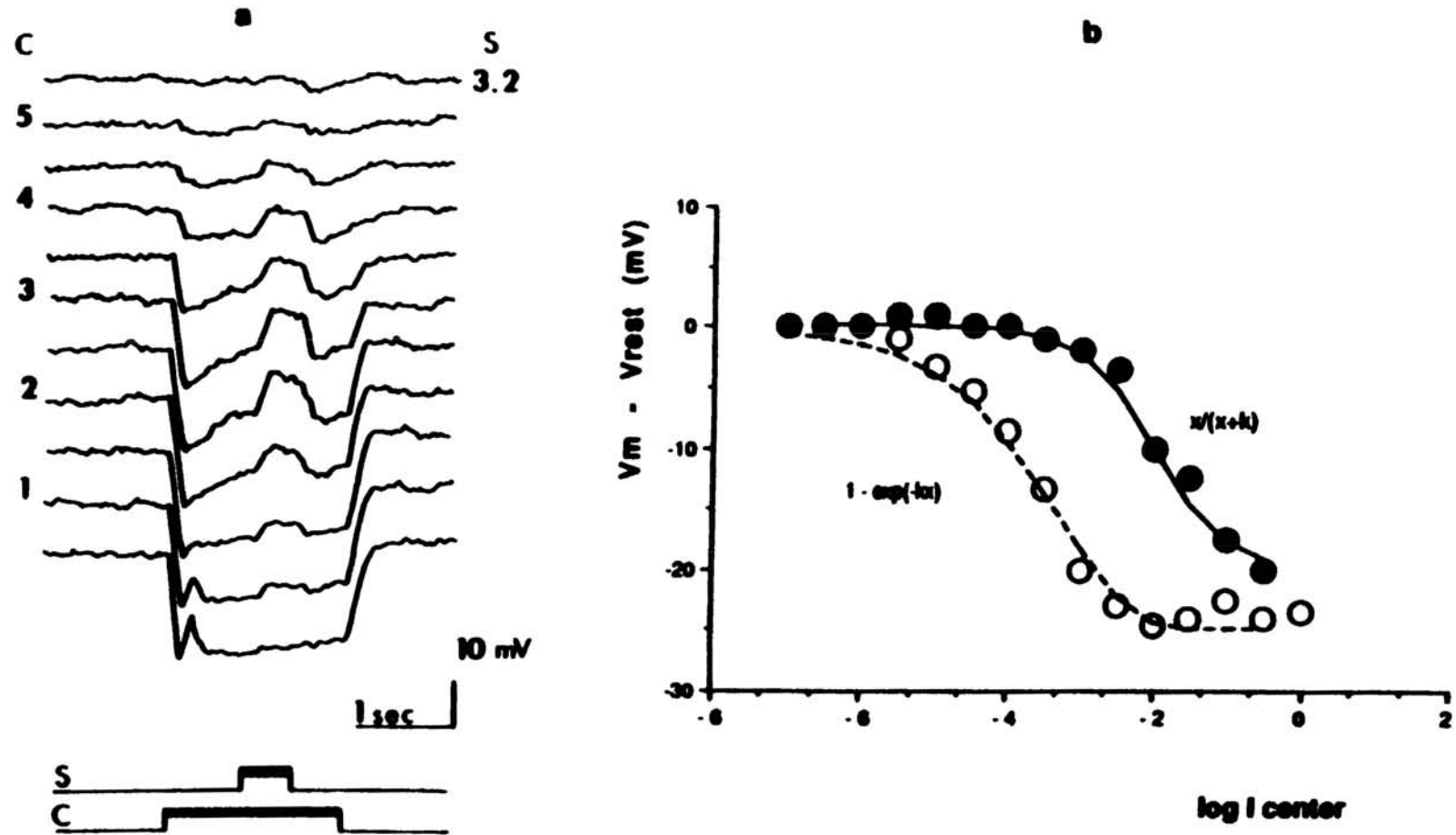

Fig. 1 (a) Series of responses to a combination of center spot and annulus. Surround illumination (S) was fixed at -3.2 l.u. throughout the experiment. Center spot intensity (C) was increased in 0.5 l.u. steps as indicated by the numbers near each trace. In the dark (upper-most trace) surround illumination had no measurable effect on the cone membrane potential. Annulus-elicited depolarizing response increased with intensity in the center up to about -3 l.u. Further increase of the spot intensity diminished the surround response. Plot of the peak hyperpolarizing response versus center spot intensity in log units in shown in (b) as open circles. It fits the dashed curve drawn according to equation 1-exp(-kx). The curve indicated by filled circles represents the membrane potential measurements taken in the middle of the depolarizing response. This data can be approximated by a continuous curve derived from x/(x+k). All membrane potential measurement are made with respect to the resting level in the dark. This result shows that in the presence of peripheral illumination, when the feedback is activated, membrane potential follows the intensity-response curve which is shifted along the Log I axis.